# Information, prediction, and query by committee

**Yoav Freund**
Computer and Information Sciences
University of California, Santa Cruz
yoav@cse.ucsc.edu

**H. Sebastian Seung**
AT&T Bell Laboratories
Murray Hill, New Jersey
seung@physics.att.com

**Eli Shamir**
Institute of Computer Science
Hebrew University, Jerusalem
shamir@cs.huji.ac.il

**Naftali Tishby**
Institute of Computer Science and
Center for Neural Computation
Hebrew University, Jerusalem
tishby@cs.huji.ac.il

## Abstract

We analyze the "query by committee" algorithm, a method for filtering informative queries from a random stream of inputs. We show that if the two-member committee algorithm achieves information gain with positive lower bound, then the prediction error decreases exponentially with the number of queries. We show that, in particular, this exponential decrease holds for query learning of thresholded smooth functions.

## 1  Introduction

For the most part, research on supervised learning has utilized a *random input* paradigm, in which the learner is both trained and tested on examples drawn at random from the same distribution. In contrast, in the *query* paradigm, the learner is given the power to ask questions, rather than just passively accept examples. What does the learner gain from this additional power? Can it attain the same prediction performance with fewer examples?

Most work on query learning has been in the *constructive* paradigm, in which the learner *constructs* inputs on which to query the teacher. For some classes of boolean functions and finite automata that are not PAC learnable from random inputs, there are algorithms that can successfully PAC learn using "membership queries"[Val84, Ang88]. Query algorithms are also known for neural network learning[Bau91]. The general relevance of these positive results is unclear, since each is specific to the learning of a particular concept class. Moreover, as shown by Eisenberg and Rivest in [ER90], constructed membership queries cannot be used to reduce the number of examples required for PAC learning. That is because *random* examples provide the learner with information not only about the correct mapping, but also about the distribution of future test inputs. This information is lacking if the learner must construct inputs.

In the statistical literature, some attempt has been made towards a more fundamental understanding of query learning, there called "sequential design of experiments."[1]. It has been suggested that the optimal experiment (query) is the one with maximal Shannon information[Lin56, Fed72, Mac92]. Similar suggestions have been made in the perceptron learning literature[KR90]. Although the use of an entropic measure seems sensible, its relationship with prediction error has remained unclear.

Understanding this relationship is a main goal of the present work, and enables us to prove a positive result about the power of queries. Our work is derived within the *query filtering* paradigm, rather than the constructive paradigm. In this paradigm, proposed by [CAL90], the learner is given access to a stream of inputs drawn at random from a distribution. The learner sees every input, but chooses whether or not to query the teacher for the label. This paradigm is realistic in contexts where it is cheap to get unlabeled examples, but expensive to label them. It avoids the problems with the constructive paradigm described in [ER90] because it gives the learner free access to the input distribution.

In [CAL90] there are several suggestions for query filters together with some empirical tests of their performance on simple problems. Seung et al.[SOS92] have suggested a filter called "query by committee," and analytically calculated its performance for some perceptron-type learning problems. For these problems, they found that the prediction error decreases exponentially fast in the number of queries. In this work we present a more complete and general analysis of query by committee, and show that such an exponential decrease is guaranteed for a general class of learning problems.

We work in a Bayesian model of concept learning[HKS91] in which the target concept $f$ is chosen from a concept class $C$ according to some prior distribution $\mathcal{P}$. The concept class consists of boolean-valued functions defined on some input space $X$. An example is an input $x \in X$ along with its label $l = f(x)$. For any set of examples, we define the *version space* to be the set of all hypotheses in $C$ that are consistent with the examples. As each example arrives, it eliminates inconsistent hypotheses, and the probability of the version space (with respect to $\mathcal{P}$) is reduced. The *instantaneous information gain* (i.i.g.) is defined as the logarithm of the ratio

of version space probabilities before and after receiving the example. In this work, we study a particular kind of learner, the Gibbs learner, which chooses a hypothesis at random from the version space. In Bayesian terms, it chooses from the posterior distribution on the concept class, which is the restriction of the prior distribution to the version space.

If an *unlabeled* input $x$ is provided, the expected i.i.g. of its label can be defined by taking the expectation with respect to the probabilities of the unknown label. The input $x$ divides the version space into two parts, those hypotheses that label it as a positive example, and those that label it negative. Let the *probability ratios* of these two parts to the whole be $\chi$ and $1 - \chi$. Then the expected i.i.g. is

$$\mathcal{H}(\chi) = -\chi \log \chi - (1 - \chi) \log(1 - \chi) \ . \tag{1}$$

The goal of the learner is to minimize its *prediction error*, its probability of error on an input drawn from the *input distribution* $\mathcal{D}$. In the case of random input learning, every input $x$ is drawn independently from $\mathcal{D}$. Since the expected i.i.g. tends to zero (see [HKS91]), it seems that random input learning is inefficient. We will analyze query construction and filtering algorithms that are designed to achieve high information gain.

The rest of the paper is organized as follows. In section 2 we exhibit query construction algorithms for the high-low game. The bisection algorithm for high-low illustrates that constructing queries with high information gain can improve prediction performance. But the failure of bisection for multi-dimensional high-low exposes a deficiency of the query construction paradigm. In section 3 we define the query filtering paradigm, and discuss the relation between information gain and prediction error for queries filtered by a committee of Gibbs learners. In section 4 lower bounds for information gain are proved for the learning of some nontrivial concept classes. Section 5 is a summary and discussion of open problems.

## 2  Query construction and the high-low game

In this section, we give examples of query construction algorithms for the high-low game and its generalizations. In the high-low game, the concept class $C$ consists of functions of the form

$$f_w(x) = \begin{cases} 1, & w < x \\ 0, & w > x \end{cases} , \tag{2}$$

where $0 \leq w, x \leq 1$. Thus both $X$ and $C$ are naturally mapped to the interval $[0, 1]$. Both $\mathcal{P}$, the prior distribution for the parameter $w$, and $\mathcal{D}$, the input distribution for $x$, are assumed to be uniform on $[0, 1]$. Given any sequence of examples, the version space is $[x_L, x_R]$ where $x_L$ is the largest negative example and $x_R$ is the smallest positive example. The posterior distribution is uniform in the interval $[x_L, x_R]$ and vanishes outside.

The prediction error of a Gibbs learner is $\Pr(f_v(x) \neq f_w(x))$ where $x$ is chosen from $\mathcal{D}$, and $v$ and $w$ from the posterior distribution. It is easy to show that $\Pr(f_v(x) \neq f_w(x)) = (x_R - x_L)/3$. Since the prediction error is proportional to the version space volume, always querying on the midpoint $(x_R + x_L)/2$ causes the prediction error after $m$ queries to decrease like $2^{-m}$. This is in contrast to the case of random input learning, for which the prediction error decreases like $1/m$.

The strategy of bisection is clearly maximally informative, since it achieves $\mathcal{H}(1/2) = 1$ bit per query, and can be applied to the learning of any concept class. Naive intuition suggests that it should lead to rapidly decreasing prediction error, but this is not necessarily so. Generalizing the high-low game to $d$ dimensions provides a simple counterexample. The target concepts are functions of the form

$$f_{\vec{w}}(i, x) = \begin{cases} 1, & w_i < x \\ 0, & w_i > x \end{cases} . \tag{3}$$

The prior distribution of $\vec{w}$ is uniform on the concept class $C = [0, 1]^d$. The inputs are pairs $(i, x)$, where $i$ takes on the values $1, \ldots, d$ with equal probability, and $x$ is uniformly distributed on $[0, 1]$. Since this is basically $d$ concurrent high-low games (one for each component of $\vec{w}$), the version space is a product of subintervals of $[0, 1]$. For $d = 2$, the concept class is the unit square, and the version space is a rectangle. The prediction error is proportional to the perimeter of the rectangle. A sequence of queries with $i = 1$ can bisect the rectangle along one dimension, yielding 1 bit per query, while the perimeter tends to a finite constant. Hence the prediction error tends to a finite constant, in spite of the maximal information gain.

## 3   The committee filter: information and prediction

The dilemma of the previous section was that constructing queries with high information gain does not necessarily lead to rapidly decreasing prediction error. This is because the constructed query distribution may have nothing to do with the input distribution $\mathcal{D}$. This deficiency can be avoided in a different paradigm in which the query distribution is created by filtering $\mathcal{D}$. Suppose that the learner receives a stream of unlabeled inputs $x_1, x_2, \ldots$ drawn independently from the distribution $\mathcal{D}$. After seeing each input $x_i$, the learner has the choice of whether or not to query the teacher for the correct label $l_i = f(x_i)$.

In [SOS92] it was suggested to filter queries that cause disagreement in a committee of Gibbs learners. In this paper we concentrate on committees with two members. The algorithm is:

### Query by a committee of two
**Repeat the following until $n$ queries have been accepted**

1. Draw an unlabeled input $x \in X$ at random from $\mathcal{D}$.

2. Select two hypotheses $h_1, h_2$ from the posterior distribution. In other words, pick two hypotheses that are consistent with the labeled examples seen so far.

3. If $h_1(x) \neq h_2(x)$ then query the teacher for the label of $x$, and add it to the training set.

The committee filter tends to select examples that split the version space into two parts of comparable size, because if one of the parts contains most of the version space, then the probability that the two hypotheses will disagree is very small. More precisely, if $x$ cuts the version space into parts of size $\chi$ and $1 - \chi$, then the probability of accepting $x$ is $2\chi(1 - \chi)$. One can show that the i.i.g. of the queries is lower bounded by that obtained from random inputs.

In this section, we assume something stronger: that the expected i.i.g. of the committee has positive lower bound. Conditions under which this assumption holds will be discussed in the next section. The bound implies that the cumulative information gain increases linearly with the number of queries $n$. But the version space resulting from *the queries alone* must be larger than the version space that would result if the learner knew *all of the labels*. Hence the cumulative information gain from the queries is upper bounded by the cumulative information gain which would be obtained from the labels of all $m$ inputs, which behaves like $O(d \log \frac{m}{d})$ for a concept class $C$ with finite VC dimension $d$ ([HKS91]). These $O(n)$ and $O(\log m)$ behaviors are consistent only if the gap between consecutive queries increases exponentially fast. This argument is depicted in Figure 1.

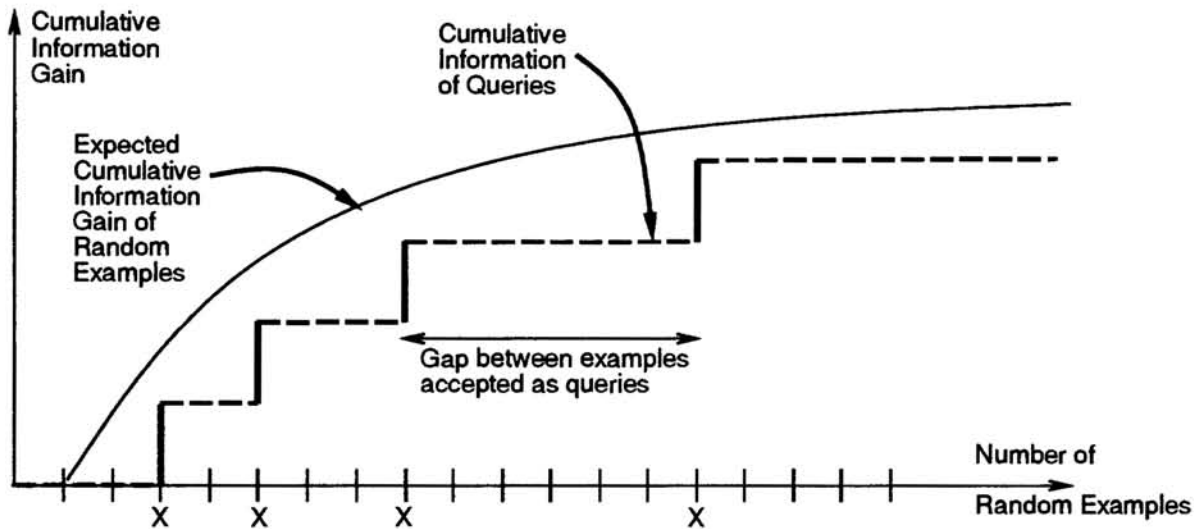

Figure 1: Each tag on the $x$ axis denotes a random example in a specific typical sequence. The symbol $X$ under a tag denotes the fact that the example was chosen as a query.

Recall that an input is accepted if it provokes disagreement between the two Gibbs learners that constitute the committee. Thus a large gap between consecutive queries is equivalent to a small *probability* of disagreement. But in our Bayesian framework the probability of disagreement between two Gibbs learners is equal to the probability of disagreement between a Gibbs learner and the teacher, which is the expected prediction error. Thus the prediction error is exponentially small as a function of the number of queries. The exact statement of the result is given below, a detailed proof of which will be published elsewhere.

**Theorem 1** *Suppose that a concept class $C$ has VC-dimension $d < \infty$ and the expected information gained by the two member committee algorithm is bounded by $c > 0$, independent of the query number and of the previous queries. Then the probability that one of the two committee members makes a mistake on a randomly chosen example with respect to a randomly chosen $f \in C$ is bounded by*

$$\left(3 + O(e^{-c_1 n})\right)\frac{n}{d}\exp\left(-\frac{c}{2(d+1)}n\right) \qquad (4)$$

*for some constant $c_1 > 0$, where $n$ is the number of queries asked so far.*

## 4    Lower bounds on the information gain

Theorem 1 is applicable to learning problems for which the committee achieves i.i.g. with positive lower bound. A simple case of this is the $d$-dimensional high-low game of section 2, for which the i.i.g. is $7/(12 \ln 2) \approx 0.84$, independent of dimension. This exact result is simple to derive because the high-low game is geometrically trivial: all version spaces are similar to each other. In general, the shape of the version space is more complex, and depends on the randomness of the examples. Nevertheless, the expected i.i.g. can be lower bounded even for some learning problems with nontrivial version space geometry.

### 4.1    The information gain for convex version spaces

Define a class of functions $f_{\vec{w}}$ by

$$f_{\vec{w}}(\vec{x}, t) = \begin{cases} 1, & \vec{w} \cdot \vec{x} > t , \\ 0, & \vec{w} \cdot \vec{x} < t . \end{cases} \tag{5}$$

The vector $\vec{w} \in \mathbf{R}^d$ is drawn at random from a prior distribution $\mathcal{P}$, which is uniform over some convex body contained in the unit ball. The distribution of inputs $(\vec{x}, t) \in B^d \times [-1, 1]$, is a product of any distribution over $B^d$ (the unit ball centered at the origin) and the uniform distribution over $[-1, +1]$. Since each example defines a plane in the concept space, all version spaces for this problem are convex. We show that there is a uniform lower bound on the expected i.i.g. for any convex version space when a two member committee filters inputs drawn from $\mathcal{D}$. In the next paragraphs we sketch our proof, the full details of which shall appear elsewhere.

In fact, we prove a stronger statement, a bound on the expected i.i.g. for any fixed $\vec{x}$. Fix $\vec{x}$ and define $\chi(t)$ as the fraction of the version space volume for which $\vec{x} \cdot \vec{w} < t$. Since the probability of filtering a query at $t$ is proportional to $2\chi(t)[1 - \chi(t)]$, the expected i.i.g. is given by

$$\mathcal{I}[\chi(t)] = \frac{\int_{-1}^{1} 2\chi(t)[1 - \chi(t)]\mathcal{H}(\chi(t))dt}{\int_{-1}^{1} 2\chi(t)[1 - \chi(t)]dt} . \tag{6}$$

In the following, it is more convenient to define the expected i.i.g. as a functional of $r(t) = \sqrt[d-1]{d\chi/dt}$, which is the radius function of the body of revolution with equivalent cross sectional area $d\chi/dt$. Using the Brunn-Minkowski inequality, it can be shown that any convex body has a concave radius function $r(t)$.

We have found a set of four transformations of $r(t)$ which decrease $\mathcal{I}[r]$. The only concave function that is a fixed point of these transformations is (up to volume preserving rescaling transformations):

$$r^*(t) = \sqrt[d-1]{d/2}\,(1 - |t|) .$$

This corresponds to the body constructed by placing two cones base to base with their axes pointing along $\vec{x}$. We can calculate $\mathcal{I}[r^*]$ explicitly for each dimension $d$. As the dimension of the space increases to infinity this value converges from above to a strictly positive value which is $1/9 + 7/(18 \ln 2) \approx 0.672$ bits, which is surprisingly close to the upper bound of 1 bit.

## 4.2    The information gain for thresholded continuous functions

Consider a concept class consisting of functions of the form

$$f_{\vec{w}}(x) = \begin{cases} 1, & F(\vec{w}, \vec{x}) \geq 0 \ , \\ 0, & F(\vec{w}, \vec{x}) < 0 \ , \end{cases} \tag{7}$$

where $\vec{x} \in \mathbf{R}^l$, $\vec{w} \in \mathbf{R}^d$ and $F$ is a smooth function of both $\vec{x}$ and $\vec{w}$. Random input learning of this type of concept class has been studied within the annealed approximation by [AFS92]. We assume that both $\mathcal{D}$ and $\mathcal{P}$ are described by density functions that are smooth and nonvanishing almost everywhere. Let the target concept be denoted by $f_{\vec{w}_0}(\vec{x})$. We now argue that in the small version space limit (reached in the limit of a large number of examples), the expected i.i.g. for query learning of this concept class has the same lower bound that was derived in section 4.1.

This is because a linear expansion of $F$ becomes a good approximation in the version space,

$$F(\vec{w}, \vec{x}) = F(\vec{w}_0, \vec{x}) + (\vec{w} - \vec{w}_0) \cdot \nabla_{\vec{w}} F(\vec{w}_0, \vec{x}) \ . \tag{8}$$

Consequently, the version space is a convex body containing $\vec{w}_0$, each boundary of which is a hyperplane perpendicular to $\nabla_w F(\vec{w}_0, \vec{x})$ for some $\vec{x}$ in the training set. Because the prior density $\mathcal{P}$ is smooth and nonvanishing, the posterior becomes uniform on the version space.

From Eq. (8) it follows that a small version space is only cut by hyperplanes corresponding to inputs $\vec{x}$ for which $F(\vec{w}_0, \vec{x})$ is small. Such inputs can be parametrized by using coordinates on the decision boundary (the manifold in $\vec{x}$ space determined by $F(\vec{w}_0, \vec{x}) = 0$), plus an additional coordinate for the direction normal to the decision boundary. Varying the normal coordinate of $\vec{x}$ changes the distance of the corresponding hyperplane from $\vec{w}_0$, but does not change its direction (to lowest order). Hence each normal average is governed by the lower bound of 0.672 bits that was derived in section 4.1 for planar cuts along a fixed axis of a convex version space. The expected i.i.g. is obtained by integrating the normal average over the rest of the coordinates, and therefore is governed by the same lower bound.

## 5    Summary and open questions

In this work we have shown that the number of examples required for query learning behaves like the logarithm of the number required for random input learning. This result on the power of query filtering applies generally to concept classes for which the committee filter achieves information gain with positive lower bound, and in particular to concept classes consisting of thresholded smooth functions. A wide variety of learning architectures in common use fall in this group, including radial basis function networks and layered feedforward neural networks with smooth transfer functions. Our main unrealistic assumption is that the learned rule is assumed to be realizable and noiseless. Understanding how to filter queries for learning unrealizable or noisy concepts remains an important open problem.

## Acknowledgments

Part of this research was done at the Hebrew University of Jerusalem. Freund, Shamir and Tishby would like to thank the US-Israel Binational Science Foundation (BSF) Grant no. 90-00189/2 for support of their work. We would also like to thank Yossi Azar and Manfred Opper for helpful discussions regarding this work.

## Footnotes

[1]The paradigm of (non-sequential) experimental design is analogous to what might be called "batch query learning," in which all of the inputs are chosen by the learner before a single label is received from the teacher

## References

[AFS92] S. Amari, N. Fujita, and S. Shinomoto. Four types of learning curves. *Neural Comput.*, 4:605–618, 1992.

[Ang88] D. Angluin. Queries and concept learning. *Machine Learning*, 2:319–342, 1988.

[Bau91] E. Baum. Neural net algorithms that learn in polynomial time from examples and queries. *IEEE Trans. Neural Networks*, 2:5–19, 1991.

[CAL90] D. Cohn, L. Atlas, and R. Ladner. Training connectionist networks with queries and selective sampling. *Advances in Neural Information Processing Systems*, 2:566–573, 1990.

[ER90] B. Eisenberg and R. Rivest. On the sample complexity of PAC-learning using random and chosen examples. In M. Fulk and J. Case, editors, *Proceedings of the Third Annual ACM Workshop on Computational Learning Theory*, pages 154–162, San Mateo, Ca, 1990. Kaufmann.

[Fed72] V. V. Fedorov. *Theory of Optimal Experiments*. Academic Press, New York, 1972.

[HKS91] D. Haussler, M. Kearns, and R. Schapire. Bounds on the sample complexity of Bayesian learning using information theory and the VC dimension. In M. K. Warmuth and L. G. Valiant, editors, *Proceedings of the Fourth Annual Workshop on Computational Learning Theory*, pages 61–74, San Mateo, CA, 1991. Kaufmann.

[KR90] W. Kinzel and P. Ruján. Improving a network generalization ability by selecting examples. *Europhys. Lett.*, 13:473–477, 1990.

[Lin56] D. V. Lindley. On a measure of the information provided by an experiment. *Ann. Math. Statist.*, 27:986–1005, 1956.

[Mac92] D. J. C. MacKay. *Bayesian methods for adaptive models*. PhD thesis, California Institute of Technology, Pasadena, 1992.

[SOS92] H. S. Seung, M. Opper, and H. Sompolinsky. Query by committee. In *Proceedings of the fifth annual ACM workshop on computational learning theory*, pages 287–294, New York, 1992. ACM.

[Val84] L. G. Valiant. A theory of the learnable. *Comm. ACM*, 27:1134–1142, 1984.
